# Computation of Similarity Measures for Sequential Data using Generalized Suffix Trees

**Konrad Rieck**
Fraunhofer FIRST.IDA
Kekuléstr. 7
12489 Berlin, Germany
rieck@first.fhg.de

**Pavel Laskov**
Fraunhofer FIRST.IDA
Kekuléstr. 7
12489 Berlin, Germany
laskov@first.fhg.de

**Sören Sonnenburg**
Fraunhofer FIRST.IDA
Kekuléstr. 7
12489 Berlin, Germany
sonne@first.fhg.de

## Abstract

We propose a generic algorithm for computation of similarity measures for sequential data. The algorithm uses generalized suffix trees for efficient calculation of various kernel, distance and non-metric similarity functions. Its worst-case run-time is linear in the length of sequences and independent of the underlying embedding language, which can cover words, $k$-grams or all contained subsequences. Experiments with network intrusion detection, DNA analysis and text processing applications demonstrate the utility of distances and similarity coefficients for sequences as alternatives to classical kernel functions.

## 1 Introduction

The ability to operate on sequential data is a vital prerequisite for application of machine learning techniques in many challenging domains. Examples of such applications are natural language processing (text documents), bioinformatics (DNA and protein sequences) and computer security (byte streams or system call traces). A key instrument for handling such data is the efficient computation of pairwise similarity between sequences. Similarity measures can be seen as an abstraction between particular structure of data and learning theory.

One of the most successful similarity measures thoroughly studied in recent years is the kernel function [e.g. 1–3]. Various kernels have been developed for sequential data, starting from the original ideas of Watkins [4] and Haussler [5] and extending to application-specific kernels such as the ones for text and natural language processing [e.g. 6–8], bioinformatics [e.g. 9–14], spam filtering [15] and computer security [e.g. 16; 17].

Although kernel-based learning has gained a major focus in machine learning research, a kernel function is obviously only one of various possibilities for measuring similarity between objects. The choice of a similarity measure is essentially determined by (a) understanding of a problem and (b) properties of the learning algorithm to be applied. Some algorithms operate in vector spaces, others in inner product, metric or even non-metric feature spaces. Investigation of techniques for learning in spaces other than RKHS is currently one of the active research fields in machine learning [e.g. 18–21].

The focus of this contribution lies on general similarity measures for sequential data, especially on efficient algorithms for their computation. A large number of such similarity measures can be expressed in a generic form so that a simple linear-time algorithm can be applied for computation of *a wide class of similarity measures*. This algorithm enables the investigation of alternative representations of problem domain knowledge other than kernel functions. As an example, two applications are presented for which replacement of a kernel – or equivalently, the Euclidean distance – with a different similarity measure yields a significant improvement of accuracy in an unsupervised learning scenario.

The rest of the paper is organized as follows. Section 2 provides a brief review of common similarity measures for sequential data and introduces a generic form in which a large variety of them can be cast. The generalized suffix tree and a corresponding algorithm for linear-time computation of similarity measures are presented in Section 3. Finally, the experiments in Section 4 demonstrate efficiency and utility of the proposed algorithm on real-world applications: network intrusion detection, DNA sequence analysis and text processing.

## 2  Similarity measures for sequences

### 2.1  Embedding of sequences

A common way to define similarity measures for sequential data is via explicit embedding into a high-dimensional feature space. A sequence $\mathbf{x}$ is defined as concatenation of symbols from a finite alphabet $\Sigma$. To model the content of a sequence, we consider a language $L \subseteq \Sigma^*$ comprising subsequences $w \in L$. We refer to these subsequences as *words*, even though they may not correspond to a natural language. Typical examples for $L$ are a "bag of words" [e.g. 22], the set of all sequences of fixed length ($k$-grams or $k$-mers) [e.g. 10; 23] or the set of all contained subsequences [e.g. 8; 24].

Given a language $L$, a sequence $\mathbf{x}$ can be mapped into an $|L|$-dimensional feature space by calculating an embedding function $\phi_w(\mathbf{x})$ for every $w \in L$ appearing in $\mathbf{x}$. The funcion $\phi_w$ is defined as follows

$$\phi_w : \Sigma^* \to \mathbb{R}^+ \cup \{0\}, \quad \phi_w(\mathbf{x}) := \psi(\mathrm{occ}(w, \mathbf{x})) \cdot \mathcal{W}_w \tag{1}$$

where $\mathrm{occ}(w, \mathbf{x})$ is the number of occurrences of $w$ in $\mathbf{x}$, $\psi$ a numerical transformation, e.g. a conversion to frequencies, and $\mathcal{W}$ a weighting assigned to individual words, e.g. length-dependent or position-dependent weights [cf. 3; 24]. By employing the feature space induced through $L$ and $\phi$, one can adapt many vectorial similarity measures to operate on sequences.

The feature space defined via explicit embedding is sparse, since the number of non-zero dimensions for each feature vector is bounded by the sequence length. Thus the essential parameter for measuring complexity of computation is the sequence length, denoted hereinafter as $n$. Furthermore, the length of a word $|w|$ or in case of a set of words the maximum length is denoted by $k$.

### 2.2  Vectorial similarity measures

Several vectorial kernel and distance functions can be applied to the proposed embedding of sequential data. A list of common functions in terms of $L$ and $\phi$ is given in Table 1.

| Kernel function | $k(\mathbf{x}, \mathbf{y})$ |
|---|---|
| Linear | $\sum_{w \in L} \phi_w(\mathbf{x}) \phi_w(\mathbf{y})$ |
| Polynomial | $\left( \sum_{w \in L} \phi_w(\mathbf{x}) \phi_w(\mathbf{y}) + \theta \right)^d$ |
| RBF | $\exp\left( \frac{-d(\mathbf{x}, \mathbf{y})^2}{\sigma} \right)$ |
| **Distance function** | $d(\mathbf{x}, \mathbf{y})$ |
| Manhattan | $\sum_{w \in L} |\phi_w(\mathbf{x}) - \phi_w(\mathbf{y})|$ |
| Canberra | $\sum_{w \in L} \frac{|\phi_w(\mathbf{x}) - \phi_w(\mathbf{y})|}{\phi_w(\mathbf{x}) + \phi_w(\mathbf{y})}$ |
| Minkowski | $\sqrt[k]{\sum_{w \in L} |\phi_w(\mathbf{x}) - \phi_w(\mathbf{y})|^k}$ |
| Hamming | $\sum_{w \in L} \mathrm{sgn} \, |\phi_w(\mathbf{x}) - \phi_w(\mathbf{y})|$ |
| Chebyshev | $\max_{w \in L} |\phi_w(\mathbf{x}) - \phi_w(\mathbf{y})|$ |

Table 1: Kernels and distances for sequential data

| Similarity coefficient | $s(\mathbf{x}, \mathbf{y})$ |
|---|---|
| Simpson | $a/\min(a+b, a+c)$ |
| Jaccard | $a/(a+b+c)$ |
| Braun-Blanquet | $a/\max(a+b, a+c)$ |
| Czekanowski, Sorensen-Dice | $2a/(2a+b+c)$ |
| Sokal-Sneath, Anderberg | $a/(a+2(b+c))$ |
| Kulczynski (1st) | $a/(b+c)$ |
| Kulczynski (2nd) | $\frac{1}{2}(a/(a+b) + a/(a+c))$ |
| Otsuka, Ochiai | $a/\sqrt{(a+b)(a+c)}$ |

Table 2: Similarity coefficients for sequential data

Beside kernel and distance functions, a set of rather exotic similarity coefficients is also suitable for application to sequential data [25]. The coefficients are constructed using three summation variables $a, b$ and $c$, which in the case of binary vectors correspond to the number of matching component pairs (1-1), left mismatching pairs (0-1) and right mismatching pairs (1-0) [cf. 26; 27] Common similarity coefficients are given in Table 2. For application to non-binary data these summation variables can be extended as proposed in [25]:

$$a = \sum_{w \in L} \min(\phi_w(\mathbf{x}), \phi_w(\mathbf{y}))$$
$$b = \sum_{w \in L} [\phi_w(\mathbf{x}) - \min(\phi_w(\mathbf{x}), \phi_w(\mathbf{y}))]$$
$$c = \sum_{w \in L} [\phi_w(\mathbf{y}) - \min(\phi_w(\mathbf{x}), \phi_w(\mathbf{y}))]$$

### 2.3 A generic representation

One can easily see that the presented similarity measures can be cast in a generic form that consists of an outer function $\oplus$ and an inner function $m$:

$$s(\mathbf{x}, \mathbf{y}) = \bigoplus_{w \in L} m(\phi_w(\mathbf{x}), \phi_w(\mathbf{y})) \tag{2}$$

Given this definition, the kernel and distance functions presented in Table 1 can be re-formulated in terms of $\oplus$ and $m$. Adaptation of similarity coefficients to the generic form (2) involves a re-formulation of the summation variables $a$, $b$ and $c$. The particular definitions of outer and inner functions for the presented similarity measures are given in Table 3. The polynomial and RBF kernels are not shown since they can be expressed in terms of a linear kernel or a distance respectively.

| Kernel function | $\oplus$ | $m(x, y)$ | Distance function | $\oplus$ | $m(x, y)$ |
|---|---|---|---|---|---|
| Linear | $+$ | $x \cdot y$ | Manhattan | $+$ | $|x - y|$ |
| **Similarity coef.** | $\oplus$ | $m(x, y)$ | Canberra | $+$ | $|x - y|/(x + y)$ |
| Variable $a$ | $+$ | $\min(x, y)$ | Minkowski$^k$ | $+$ | $|x - y|^k$ |
| Variable $b$ | $+$ | $x - \min(x, y)$ | Hamming | $+$ | $\operatorname{sgn}|x - y|$ |
| Variable $c$ | $+$ | $y - \min(x, y)$ | Chebyshev | $\max$ | $|x - y|$ |

Table 3: Generalized formulation of similarity measures

# 3    Generalized suffix trees for comparison of sequences

The key to efficient comparison of two sequences lies in considering only the minimum of words necessary for computation of the generic form (2) of similarity measures. In the case of kernels only the *intersection of words* in both sequences needs to be considered, while the *union of words* is needed for calculating distances and non-metric similarity coefficients. A simple and well-known approach for such comparison is representing the words of each sequence in a sorted list. For words of maximum length $k$ such a list can be constructed in $O(kn \log n)$ using general sorting or $O(kn)$ using radix-sort. If the length of words $k$ is unbounded, sorted lists are no longer an option as the sorting time becomes quadratic.

Thus, special data structures are needed for efficient comparison of sequences. Two data structures previously used for computation of kernels are tries [28; 29] and suffix trees [30]. Both have been applied for computation of a variety of kernel functions in $O(kn)$ [3; 10] and also in $O(n)$ run-time using matching statistics [24]. In this contribution we will argue that a generalized suffix tree is suitable for computation of *all similarity measures* of the form (2) in $O(n)$ run-time.

A *generalized suffix tree* (GST) is a tree containing all suffixes of a set of strings $\mathbf{x}_1, \ldots, \mathbf{x}_l$ [31]. The simplest way to construct a generalized suffix tree is to extend each string $\mathbf{x}_i$ with a delimiter $\$_i$ and to apply a suffix tree construction algorithm [e.g. 32] to the concatenation of strings $\mathbf{x}_1\$_1 \ldots \mathbf{x}_l\$_l$. In the remaining part we will restrict ourselves to the case of two strings $\mathbf{x}$ and $\mathbf{y}$ delimited by # and \$, computation of an entire similarity matrix using a single GST for a set of strings being a straightforward extension. An example of a generalized suffix tree for the strings "aab#" and "babab\$" is shown in Fig. 1(a).

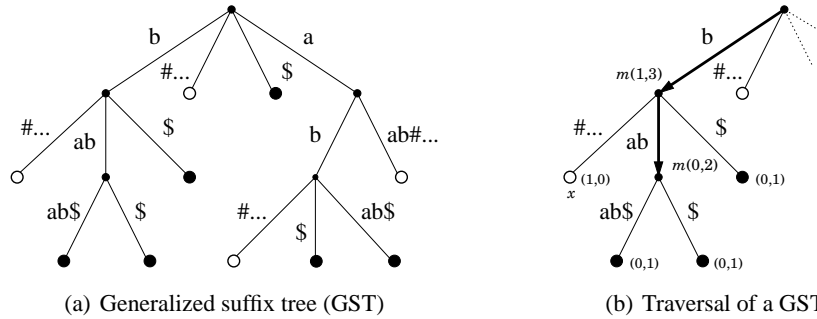

(a) Generalized suffix tree (GST)          (b) Traversal of a GST

Figure 1: Generalized suffix tree for "aab#" and "babab\$" and a snapshot of its traveral

Once a generalized suffix tree is constructed, it remains to determine the number of occurences $\mathrm{occ}(w, \mathbf{x})$ and $\mathrm{occ}(w, \mathbf{y})$ of each word $w$ present in the sequences $\mathbf{x}$ and $\mathbf{y}$. Unlike the case for kernels for which only nodes corresponding to both sequences need to be considered [24], the contributions must be correctly computed for *all* nodes in the generalized suffix tree. The following simple recursive algorithm computes a generic similarity measure between the sequence $\mathbf{x}$ and $\mathbf{y}$ in one depth-first traversal of the generalized suffix tree (cf. Algorithm 1).

The algorithm exploits the fact that a leaf in a GST representing a suffix of $\mathbf{x}$ contributes exactly 1 to $\mathrm{occ}(w, \mathbf{x})$ if $w$ is the prefix of this suffix – and similarly for $\mathbf{y}$ and $\mathrm{occ}(w, \mathbf{y})$. As the GST contains all suffixes of $\mathbf{x}$ and $\mathbf{y}$, every word $w$ in $\mathbf{x}$ and $\mathbf{y}$ is represented by at least one leaf. Whether a leaf contributes to $\mathbf{x}$ or $\mathbf{y}$ can be determined by considering the edge at the leaf. Due to the uniqueness of the delimiter #, no branching nodes can occur below an edge containing #, thus a leaf node at an edge starting before the index of # must contain a suffix of $\mathbf{x}$; otherwise it contains a suffix of $\mathbf{y}$. The contributions of all leaves are aggregated in two variables $x$ and $y$ during a post-order traversal. At each node the inner function $m$ of (2) is calculated using $\psi(x)$ and $\psi(y)$ according to the embedding $\phi$ in (1). A snapshot of the traversal procedure is illustrated in Fig. 1(b).

To account implicit nodes along the edges of the GST and to support weighted embeddings $\phi$, the weighting function WEIGHT introduced in [24] is employed. At a node $v$ the function takes the beginning ($begin[v]$) and the end ($end[v]$) of the incoming edge and the depth of node ($depth[v]$) as arguments to determine how much the node and edge contribute to the similarity measure, e.g. for $k$-gram models only nodes up to a path depth of $k$ need to be considered.

---

**Algorithm 1** Suffix tree comparison

---
1: **function** COMPARE($\mathbf{x}, \mathbf{y}$)
2:      $S \leftarrow$ SUFFIXTREE($\mathbf{x} \# \mathbf{y}$ \$)
3:      $(x, y, s) \leftarrow$ MATCH($root[S]$)
4:      **return** $s$
5:
6: **function** MATCH($v$)
7:      **if** $v$ is leaf **then**
8:          $s \leftarrow 0$
9:          **if** $begin[v] \leq index_\#$ **then**
10:             $(x, y) \leftarrow (1, 0)$          ▷ Leaf of a suffix of $\mathbf{x}$
11:             $j \leftarrow index_\# - 1$
12:         **else**
13:             $(x, y) \leftarrow (0, 1)$          ▷ Leaf of a suffix of $\mathbf{y}$
14:             $j \leftarrow index_\$ - 1$
15:     **else**
16:         $(x, y, s) \leftarrow (0, 0, 0)$
17:         **for all** $c$ in $children[v]$ **do**
18:             $(\hat{x}, \hat{y}, \hat{s}) \leftarrow$ MATCH($c$)          ▷ Traverse GST
19:             $(x, y, s) \leftarrow (x + \hat{x}, y + \hat{y}, s \oplus \hat{s})$
20:         $j \leftarrow end[v]$
21:     $\mathcal{W} \leftarrow$ WEIGHT($begin[v], j, depth[v]$)
22:     $s \leftarrow s \oplus m(\psi(x)\mathcal{W}, \psi(y)\mathcal{W})$          ▷ Cf. definitions in (1) and (2)
23:     **return** $(x, y, s)$

---

Similarly to the extension of string kernels proposed in [33], the GST traversal can be performed on an enhanced suffix array [34] for further run-time and space reduction.

To prove correctness of our algorithm, a different approach must be taken than the one in [24]. We cannot claim that the computed similarity value is equivalent to the one returned by the matching statistic algorithm, since the latter is restricted to kernel functions. Instead we show that at each recursive call to the MATCH function correct numbers of occurences are maintained.

**Theorem 1.** *A word $w$ occurs* $occ(w, \mathbf{x})$ *and* $occ(w, \mathbf{y})$ *times in $\mathbf{x}$ and $\mathbf{y}$ if and only if* MATCH($\bar{w}$) *returns $x = occ(w, \mathbf{x})$ and $y = occ(w, \mathbf{y})$, where $\bar{w}$ is the node at the end of a path from the root reassembling $w$ in the generalized suffix tree of $\mathbf{x}$ and $\mathbf{y}$ .*

*Proof.* If $w$ occurs $m$ times in $\mathbf{x}$, there exist exactly $m$ suffixes of $\mathbf{x}$ with $w$ as prefix. Since $w$ corresponds to a path from the root of the GST to a node $\bar{w}$ all $m$ suffixes must pass $\bar{w}$. Due to the unique delimiters $\#$ each suffix of $\mathbf{x}$ corresponds to one leaf node in the GST whose incoming edge contains $\#$. Hence $m$ equals $occ(w, \mathbf{x})$ and is exactly the aggregated quantity $x$ returned by MATCH($\bar{w}$). Likewise, $occ(w, \mathbf{y})$ is the number of suffixes beginning after $\#$ and having a prefix $w$, which is computed by $y$.     □

## 4   Experimental Results

### 4.1   Run-time experiments

In order to illustrate the efficiency of the proposed algorithm, we conducted run-time experiments on three benchmark data sets for sequential data: network connection payloads from the DARPA 1999 IDS evaluation [35], news articles from the Reuters-21578 data set [36] and DNA sequences from the human genome [14]. Table 4 gives an overview of the data sets and their specific properties. We compared the run-time of the generalized suffix tree algorithm with a recent trie-based method supporting computation of distances. Tries yield better or equal run-time complexity for computation of similarity measures over $k$-grams than algorithms using indexed arrays and hash tables. A detailed description of the trie-based approach is given in [25]. Note that in all of the following experiments tries were generated in a pre-processing step and the reported run-time corresponds to the comparison procedure only.

For each of the three data sets, we implemented the following experimental protocol: the Manhattan distances were calculated for 1000 pairs of randomly selected sequences using $k$-grams as an em-

| Name | Type | Alphabet | Min. length | Max. length |
|------|------|----------|-------------|-------------|
| DNA | Human genome sequences | 4 | 2400 | 2400 |
| NIDS | TCP connection payloads | 108 | 53 | 132753 |
| TEXT | Reuters Newswire articles | 93 | 43 | 10002 |

Table 4: Sequential data sets

bedding language. The procedure was repeated 10 times for various values of $k$, and the run-time was averaged over all runs. Fig. 2 compares the run-time of sequence comparison algorithms using the generalized suffix trees and tries. On all three data sets the trie-based comparison has a low run-time for small values of $n$ but grows linearly with $k$. The algorithm using a generalized suffix tree is independent from complexity of the embedding language, although this comes at a price of higher constants due to a more complex data structure. It is obvious that a generalized suffix tree is the algorithm of choice for higher values of $k$.

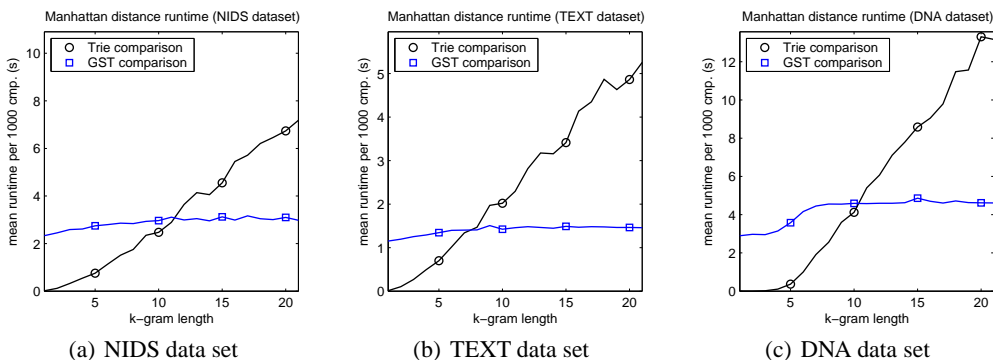

(a) NIDS data set          (b) TEXT data set          (c) DNA data set

Figure 2: Run-time performance for varying $k$-gram lengths

## 4.2 Applications

As a second part of our evaluation, we show that the ability of our approach to compute diverse similarity measures pays off when it comes to real applications, especially in an unsupervised learning scenario. The experiments were performed for (a) intrusion detection in real network traffic and (b) transcription start site (TSS) recognition in DNA sequences.

For the first application, network data was generated by members of our laboratory using virtual network servers. Recent attacks were injected by a penetration-testing expert. The distance-based anomaly detection method Zeta [17] was applied to 5-grams extracted from byte sequences of TCP connections using different similarity measures: the linear kernel, the Manhattan distance and the Kulczynski coefficient. The results on network data from the HTTP protocol are shown in Fig. 3(a). Application of the Kulczynski coefficient yields the highest detection accuracy. Over 78% of all attacks are identified with no false-positives in an unsupervised setup. In comparison, the linear kernel yields roughly 30% lower detection rates.

The second application focused on TSS recognition in DNA sequences. The data set comprises fixed length DNA sequences that either cover the TSS of protein coding genes or have been extracted randomly from the interior of genes [14]. We evaluated three methods on this data: an unsupervised $k$-nearest neighbor (kNN) classifier, a supervised and bagged kNN classifier and a Support Vector Machine (SVM). Each method was trained and tested using a linear kernel and the Manhattan distance as a similarity measure over 4-grams. Fig. 3(b) shows the performance achieved by the unsupervised and supervised versions of the kNN classifier[1]. Even though the linear kernel and the Manhattan distance yield similar accuracy in a supervised setup, their performance differs significantly in unsupervised application. In the absence of prior knowledge of labels the Manhattan

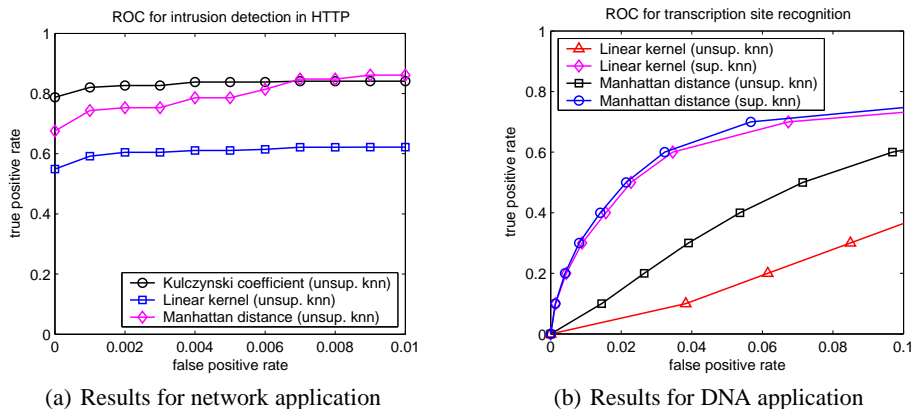

<div align="center">(a) Results for network application      (b) Results for DNA application</div>

<div align="center">Figure 3: Comparison of similarity measures on the network and DNA data</div>

distance expresses better discriminative properties for TSS recognition than the linear kernel. For the supervised application the classication performance is bounded for both similarity measures, since only some discriminative features for TSS recognition are encapsulated in $n$-gram models [14].

## 5 Conclusions

Kernel functions for sequences have recently gained strong attention in many applications of machine learning, especially in bioinformatics and natural language processing. In this contribution we have shown that other similarity measures such as metric distances or non-metric similarity coefficients can be computed with the same run-time complexity as kernel functions. The proposed algorithm is based on a post-order traversal of a generalized suffix tree of two or more sequences. During the traversal, the counts of matching and mismatching words from an embedding language are computed in time linear in sequence length – regardless of the particular kind of chosen language: words, $k$-grams or even all consecutive subsequences. By using a generic representation of the considered similarity measures based on an outer and inner function, the same algorithm can be applied for various kernel, distance and similarity functions on sequential data.

Our experiments demonstrate that the use of general similarity measures can bring significant improvement to learning accuracy – in our case observed for unsupervised learning – and emphasize importance of further investigation of distance- and similarity-based learning algorithms.

### Acknowledgments

The authors gratefully acknowledge the funding from *Bundesministerium für Bildung und Forschung* under the project MIND (FKZ 01-SC40A) and would like to thank Klaus-Robert Müller and Mikio Braun for fruitful discussions and support.

## Footnotes

[1]Results for the SVM are similar to the supervised kNN and have been omitted.

## References

[1] V.N. Vapnik. *Statistical Learning Theory*. Wiley, New York, 1998.

[2] B. Schölkopf and A.J. Smola. *Learning with Kernels*. MIT Press, Cambridge, MA, 2002.

[3] J. Shawe-Taylor and N. Cristianini. *Kernel methods for pattern analysis*. Cambridge University Press, 2004.

[4] C. Watkins. Dynamic alignment kernels. In A.J. Smola, P.L. Bartlett, B. Schölkopf, and D. Schuurmans, editors, *Advances in Large Margin Classifiers*, pages 39–50, Cambridge, MA, 2000. MIT Press.

[5] D. Haussler. Convolution kernels on discrete structures. Technical Report UCSC-CRL-99-10, UC Santa Cruz, July 1999.

[6] T. Joachims. Text categorization with support vector machines: Learning with many relevant features. Technical Report 23, LS VIII, University of Dortmund, 1997.

[7] E. Leopold and J. Kindermann. Text categorization with Support Vector Machines. how to represent texts in input space? *Machine Learning*, 46:423–444, 2002.

[8] H. Lodhi, C. Saunders, J. Shawe-Taylor, N. Cristianini, and C. Watkins. Text classification using string kernels. *Journal of Machine Learning Research*, 2:419–444, 2002.

[9] A. Zien, G. Rätsch, S. Mika, B. Schölkopf, T. Lengauer, and K.-R. Müller. Engineering Support Vector Machine Kernels That Recognize Translation Initiation Sites. *BioInformatics*, 16(9):799–807, September 2000.

[10] C. Leslie, E. Eskin, and W.S. Noble. The spectrum kernel: A string kernel for SVM protein classification. In *Proc. Pacific Symp. Biocomputing*, pages 564–575, 2002.

[11] C. Leslie, E. Eskin, A. Cohen, J. Weston, and W.S. Noble. Mismatch string kernel for discriminative protein classification. *Bioinformatics*, 1(1):1–10, 2003.

[12] J. Rousu and J. Shawe-Taylor. Efficient computation of gapped substring kernels for large alphabets. *Journal of Machine Leaning Research*, 6:1323–1344, 2005.

[13] G. Rätsch, S. Sonnenburg, and B. Schölkopf. RASE: recognition of alternatively spliced exons in c. elegans. *Bioinformatics*, 21:i369–i377, June 2005.

[14] S. Sonnenburg, A. Zien, and G. Rätsch. ARTS: Accurate Recognition of Transcription Starts in Human. *Bioinformatics*, 22(14):e472–e480, 2006.

[15] H. Drucker, D. Wu, and V.N. Vapnik. Support vector machines for spam categorization. *IEEE Transactions on Neural Networks*, 10(5):1048–1054, 1999.

[16] E. Eskin, A. Arnold, M. Prerau, L. Portnoy, and S. Stolfo. *Applications of Data Mining in Computer Security*, chapter A geometric framework for unsupervised anomaly detection: detecting intrusions in unlabeled data. Kluwer, 2002.

[17] K. Rieck and P. Laskov. Detecting unknown network attacks using language models. In *Proc. DIMVA*, pages 74–90, July 2006.

[18] T. Graepel, R. Herbrich, P. Bollmann-Sdorra, and K. Obermayer. Classification on pairwise proximity data. In M.S. Kearns, S.A. Solla, and D.A. Cohn, editors, *Advances in Neural Information Processing Systems*, volume 11, pages 438–444. MIT Press, 1999.

[19] V. Roth, J Laub, M. Kawanabe, and J.M. Buhmann. Optimal cluster preserving embedding of non-metric proximity data. *IEEE Trans. PAMI*, 25:1540–1551, December 2003.

[20] J. Laub and K.-R. Müller. Feature discovery in non-metric pairwise data. *Journal of Machine Learning*, 5(Jul):801–818, July 2004.

[21] C. Ong, X. Mary, S. Canu, and A.J. Smola. Learning with non-positive kernels. In *Proc. ICML*, pages 639–646, 2004.

[22] G. Salton. Mathematics and information retrieval. *Journal of Documentation*, 35(1):1–29, 1979.

[23] M. Damashek. Gauging similarity with $n$-grams: Language-independent categorization of text. *Science*, 267(5199):843–848, 1995.

[24] S.V.N. Vishwanathan and A.J. Smola. *Kernels and Bioinformatics*, chapter Fast Kernels for String and Tree Matching, pages 113–130. MIT Press, 2004.

[25] K. Rieck, P. Laskov, and K.-R. Müller. Efficient algorithms for similarity measures over sequential data: A look beyond kernels. In *Proc. DAGM*, pages 374–383, September 2006.

[26] R.R. Sokal and P.H. Sneath. *Principles of numerical taxonomy*. Freeman, San Francisco, CA, USA, 1963.

[27] M.R. Anderberg. *Cluster Analysis for Applications*. Academic Press, Inc., New York, NY, USA, 1973.

[28] E. Fredkin. Trie memory. *Communications of ACM*, 3(9):490–499, 1960.

[29] D. Knuth. *The art of computer programming*, volume 3. Addison-Wesley, 1973.

[30] P. Weiner. Linear pattern matching algorithms. In *Proc. 14th Annual Symposium on Switching and Automata Theory*, pages 1–11, 1973.

[31] D. Gusfield. *Algorithms on strings, trees, and sequences*. Cambridge University Press, 1997.

[32] E. Ukkonen. Online construction of suffix trees. *Algorithmica*, 14(3):249–260, 1995.

[33] C.H. Teo and S.V.N. Vishwanathan. Fast and space efficient string kernels using suffix arrays. In *Proceedings, 23rd ICMP*, pages 939–936. ACM Press, 2006.

[34] M.I. Abouelhoda, S. Kurtz, and E. Ohlebusch. Replacing suffix trees with enhanced suffix arrays. *Journal of Discrete Algorithms*, 2(1):53–86, 2002.

[35] R. Lippmann, J.W. Haines, D.J. Fried, J. Korba, and K. Das. The 1999 DARPA off-line intrusion detection evaluation. *Computer Networks*, 34(4):579–595, 2000.

[36] D.D. Lewis. Reuters-21578 text categorization test collection. AT&T Labs Research, 1997.
